# Maximum Margin Semi-Supervised Learning for Structured Variables

**Y. Altun, D. McAllester**
TTI at Chicago
Chicago, IL 60637
altun,mcallester@tti-c.org

**M. Belkin**
Department of Computer Science
University of Chicago
Chicago, IL 60637
misha@cs.uchicago.edu

## Abstract

Many real-world classification problems involve the prediction of multiple inter-dependent variables forming some structural dependency. Recent progress in machine learning has mainly focused on supervised classification of such structured variables. In this paper, we investigate structured classification in a semi-supervised setting. We present a discriminative approach that utilizes the intrinsic geometry of input patterns revealed by unlabeled data points and we derive a maximum-margin formulation of semi-supervised learning for structured variables. Unlike transductive algorithms, our formulation naturally extends to new test points.

## 1 Introduction

Discriminative methods, such as Boosting and Support Vector Machines have significantly advanced the state of the art for classification. However, traditionally these methods do not exploit dependencies between class labels where more than one label is predicted. Many real-world classification problems, on the other hand, involve sequential or structural dependencies between multiple labels. For example labeling the words in a sentence with their part-of-speech tags involves sequential dependency between part-of-speech tags; finding the parse tree of a sentence involves a structural dependency among the labels in the parse tree. Recently, there has been a growing interest in generalizing kernel methods to predict structured and inter-dependent variables in a supervised learning setting, such as dual perceptron [7], SVMs [2, 15, 14] and kernel logistic regression [1, 11]. These techniques combine the efficiency of dynamic programming methods with the advantages of the state-of-the-art learning methods. In this paper, we investigate classification of structured objects in a semi-supervised setting.

The goal of semi-supervised learning is to leverage the learning process from a small sample of labeled inputs with a large sample of unlabeled data. This idea has recently attracted a considerable amount of interest due to ubiquity of unlabeled data. In many applications from data mining to speech recognition it is easy to produce large amounts of unlabeled data, while labeling is often manual and expensive. This is also the case for many structured classification problems. A variety

of methods ranging from Naive Bayes [12], Cotraining [4], to Transductive SVM [9] to Cluster Kernels [6] and graph-based approaches [3] and references therein, have been proposed. The intuition behind many of these methods is that the classification/regression function should be smooth with respect to the geometry of the data, i. e. the labels of two inputs $x$ and $\bar{x}$ are likely to be the same if $x$ and $\bar{x}$ are similar. This idea is often represented as the *cluster assumption* or the *manifold assumption*. The unlabeled points reveal the intrinsic structure, which is then utilized by the classification algorithm. A discriminative approach to semi-supervised learning was developed by Belkin, Sindhwani and Niyogi [3, 13], where the Laplacian operator associated with unlabeled data is used as an additional penalty (regularizer) on the space of functions in a Reproducing Kernel Hilbert Space. The additional regularization from the unlabeled data can be represented as a new kernel — a "graph regularized" kernel.

In this paper, building on [3, 13], we present a discriminative semi-supervised learning formulation for problems that involve structured and inter-dependent outputs and give experimental results on max-margin semi-supervised structured classification using graph-regularized kernels. The solution of the optimization problem that utilizes both labeled and unlabeled data is a linear combination of the graph regularized kernel evaluated at the *parts* of the *labeled inputs only*, leading to a large reduction in the number of parameters. It is important to note that our classification function is defined on all input points whereas some previous work is only defined for the input points in the (labeled and unlabeled) training sample, as they use standard graph kernels, which are restricted to in-sample data points by definition.

There is an the extensive literature on semi-supervised learning and the growing number of studies on learning structured and inter-dependent variables. Delaleau et. al. [8] propose a semi-supervised learning method for standard classification that extends to out-of-sample points. Brefeld et. al. [5] is one of the first studies investigating semi-supervised structured learning problem in a discriminative framework. The most relevant previous work is the transductive structured learning proposed by Lafferty et. al. [11].

## 2 Supervised Learning for Structured Variables

In structured learning, the goal is to learn a mapping $h : \mathcal{X} \rightarrow \mathcal{Y}$ from *structured* inputs to *structured* response values, where the inputs and response values form a dependency structure. For each input $x$, there is a set of feasible outputs, $\mathcal{Y}(x) \subseteq \mathcal{Y}$. For simplicity, let us assume that $\mathcal{Y}(x)$ is finite for all $x \in \mathcal{X}$, which is the case in many real world problems and in all our examples. We denote the set of feasible input-output pairs by $\mathcal{Z} \subseteq \mathcal{X} \times \mathcal{Y}$.

It is common to construct a discriminant function $F : \mathcal{Z} \rightarrow \Re$ which maps the feasible input-output pairs to a compatibility score of the pair. To make a prediction for $x$, this score is maximized over the set of feasible outputs,

$$h(x) = \operatorname*{argmax}_{y \in \mathcal{Y}(x)} F(x, y). \tag{1}$$

The score of an $\langle x, y \rangle$ pair is computed from local fragments, or "parts", of $\langle x, y \rangle$. In Markov random fields, $x$ is a graph, $y$ is a labeling of the nodes of $x$ and a local fragment (a part) of $\langle x, y \rangle$ is a clique in $x$ and its labeling $y$. In parsing with probabilistic context free grammars, a local fragment (a part) of $\langle x, y \rangle$ consist of a branch of the tree $y$, where a branch is an internal node in $y$ together with its

children, plus all pairs of a leaf node in $y$ with the word in $x$ labeled by that node. Note that a given branch structure, such as NP $\rightarrow$ Det N, can occur more than once in a given parse tree.

In general, we let $\mathcal{P}$ be a set of (all possible) parts. We assume a "counting function", $c$, such that for $p \in \mathcal{P}$ and $\langle x, y \rangle \in Z$, $c(p, \langle x, y \rangle)$ gives the number of times that the part $p$ occurs in the pair $\langle x, y \rangle$ (the count of $p$ in $\langle x, y \rangle$). For a Mercer kernel $k : \mathcal{P} \times \mathcal{P} \rightarrow \Re$ on $\mathcal{P}$, there is an associated RHKS $\mathcal{H}_k$ of functions $f : \mathcal{P} \rightarrow \Re$, where $f$ measures the *goodness* of a part $p$. For any $f \in \mathcal{H}_k$, we define a function $F_f$ on $\mathcal{Z}$ as

$$F_f(x,y) = \sum_{p \in \mathcal{P}} c(p, \langle x, y \rangle) f(p). \tag{2}$$

Consider a simple chain example. Let $\Gamma$ be a set of possible observations and $\Sigma$ be a set of possible hidden states. We take the input $x$ to be a sequence $x_1, \ldots, x_\ell$ with $x_i \in \Gamma$ and we take $\mathcal{Y}(x)$ to be the set of all sequences $y_1, \ldots, y_\ell$ with the same length as $x$ and with $y_i \in \Sigma$. We can take $\mathcal{P}$ to be the set of all pairs $\langle s, \bar{s} \rangle$ plus all pairs $\langle s, u \rangle$ with $s, \bar{s} \in \Sigma$ and $u \in \Gamma$. Often $\Sigma$ is taken to be a finite set of "states" and $\Gamma = \Re^d$ is a set of possible feature vectors. $k(p, p')$ is commonly defined as

$$k(\langle s, \bar{s} \rangle, \langle s', \bar{s}' \rangle) \;\; = \;\; \delta(s, s') \delta(\bar{s}, \bar{s}'), \tag{3}$$
$$k(\langle s, u \rangle, \langle s', u' \rangle) \;\; = \;\; \delta(s, s') k_o(u, u'), \tag{4}$$

where $\delta(w, w')$ denotes the Kronecker-$\delta$. Note that in this example there are two types of parts — pairs of hidden states and pairs of a hidden state and an observation. Here we take $k(p, p')$ to be 0 if $p$ and $p'$ are of different types.

In the supervised learning scenario, we are given a sample $S$ of $\ell$ pairs $(\langle x^1, y^1 \rangle, \ldots, \langle x^\ell, y^\ell \rangle)$ drawn i. i. d. from an unknown but fixed probability distribution $P$ on $\mathcal{Z}$. The goal is to learn a function $f$ on the local parts $\mathcal{P}$ with small expected loss $E_P[\mathcal{L}(x, y, f)]$ where $\mathcal{L}$ is a prescribed loss function. This is commonly realized by learning $f$ that minimizes the regularized loss functional

$$f^* = \operatorname*{argmin}_{f \in \mathcal{H}_k} \sum_{i=1}^{\ell} \mathcal{L}(x^i, y^i, f) + \lambda \|f\|_k^2, \tag{5}$$

where $\|.\|_k$ is the norm corresponding to $\mathcal{H}_k$ measuring the complexity of $f$. A variety of loss functions $\mathcal{L}$ have been considered in the literature. In kernel conditional random fields (CRFs) [11], the loss function is given by

$$\mathcal{L}(x, y, f) \;\; = \;\; -F_f(x, y) + \log \sum_{\hat{y} \in \mathcal{Y}(x)} \exp(F_f(x, \hat{y}))$$

In structured Support Vector Machines (SVM), the loss function is given by

$$\mathcal{L}(x, y, f) \;\; = \;\; \max_{\hat{y} \in \mathcal{Y}(x)} \Delta(x, y, \hat{y}) + F_f(x, \hat{y}) - F_f(x, y), \tag{6}$$

where $\Delta(x, y, \hat{y})$ is some measure of distance between $y$ and $\hat{y}$ for a given observation $x$. A natural choice for $\Delta$ is to take $\Delta(x, y, \hat{y})$ to be the indicator $1_{[y \neq \hat{y}]}$ [2]. Another choice is to take $\Delta(x, y, \hat{y})$ to be the size of the symmetric difference between the sets $\mathcal{P}(\langle x, y \rangle)$ and $\mathcal{P}(\langle x, \hat{y} \rangle)$ [14].

Let $\mathcal{P}(x) \subseteq \mathcal{P}$ be the set of parts having nonzero count in some pair $\langle x, y \rangle$ for $y \in \mathcal{Y}(x)$. Let $\mathcal{P}(S)$ be the union of all sets $\mathcal{P}(x^i)$ for $x^i$ in the sample. Then, we have following straightforward variant of the Representer Theorem [10], which was also presented in [11].

**Definition:** A loss $\mathcal{L}$ is *local* if $\mathcal{L}(x, y, f)$ is determined by the value of $f$ on the set $\mathcal{P}(x)$, i.e., for $f, g : \mathcal{P} \to \Re$ we have that if $f(p) = g(p)$ for all $p \in \mathcal{P}(x)$ then $\mathcal{L}(x, y, f) = \mathcal{L}(x, y, g)$.

**Theorem 1.** *For any local loss function $\mathcal{L}$ and sample $S$ there exist weights $\alpha_p$ for $p \in \mathcal{P}(S)$ such that $f^*$ as defined by (5) can be written as follows.*

$$f^*(p) = \sum_{p' \in \mathcal{P}(S)} \alpha_{p'} k(p', p) \tag{7}$$

Thus, even though the set of feasible outputs for $x$ generally scales exponentially with the size of output, the solution can be represented in terms of the parts of the sample, which commonly scales polynomially. This is true for any loss function that partitions into parts, which is the case for loss functions discussed above.

## 3 A Semi-Supervised Learning Approach to Structured Variables

In semi-supervised learning, we are given a sample $S$ consisting of $l$ input-output pairs $\{(x^1, y^1), \ldots, (x^\ell, y^\ell)\}$ drawn i. i. d. from the probability distribution $P$ on $\mathcal{Z}$ and $u$ unlabeled input patterns $\{x^{\ell+1}, \ldots, x^{\ell+u}\}$ drawn i. i. d from the marginal distribution $P_\mathcal{X}$, where usually $l < u$. Let $\mathcal{X}(S)$ be the set $\{x^1, \ldots, x^{\ell+u}\}$ and let $\mathcal{Z}(S)$ be the set of all pairs $\langle x, y \rangle$ with $x \in \mathcal{X}(S)$ and $y \in \mathcal{Y}(x)$.

If the true classification function is smooth wrt the underlying marginal distribution, one can utilize unlabeled data points to favor functions that are smooth in this sense. Belkin et. al. [3] implement this assumption by introducing a new regularizer to the standard RHKS optimization framework (as opposed to introducing a new kernel as discussed in Section 5)

$$f^* = \operatorname*{argmin}_{f \in \mathcal{H}_k} \sum_{i=1}^{\ell} \mathcal{L}(x^i, y^i, f) + \lambda_1 ||f||_k^2 + \lambda_2 ||f||_{k_S}^2, \tag{8}$$

where $k_S$ is a kernel representing the intrinsic measure of the marginal distribution. Sindhwani et. al.[13] prove that the minimizer of (8) is in the span of a new kernel function (details below) evaluated at labeled data only. Here, we generalize this framework to structured variables and give a simplified derivation of the new kernel.

The smoothness assumption in the structured setting states that $f$ should be smooth on the underlying density on the parts $\mathcal{P}$, thus we enforce $f$ to assign similar *goodness* scores to two parts $p$ and $p'$, if $p$ and $p'$ are *similar*, for all parts of $\mathcal{Z}(S)$. Let $\mathcal{P}(S)$ be the union of all sets $\mathcal{P}(z)$ for $z \in \mathcal{Z}(S)$ and let $W$ be symmetric matrix where $W_{p,p'}$ represents the similarity of $p$ and $p'$ for $p, p' \in \mathcal{P}(S)$.

$$
\begin{aligned}
f^* &= \operatorname*{argmin}_{f \in \mathcal{H}_k} \sum_{i=1}^{\ell} \mathcal{L}(x^i, y^i, f) + \lambda_1 ||f||_k^2 + \lambda_2 \sum_{p,p' \in \mathcal{P}(S)} W_{p,p'}(f(p) - f(p'))^2 \\
&= \operatorname*{argmin}_{f \in \mathcal{H}_k} \sum_{i=1}^{\ell} \mathcal{L}(x^i, y^i, f) + \lambda_1 ||f||_k^2 + \lambda_2 \mathbf{f}^T L \mathbf{f}
\end{aligned}
\tag{9}
$$

Here $W$ is a similarity matrix (like a nearest neighbor graph) and $L$ is the Laplacian of $W$, $L = D - W$, where $D$ is a diagonal matrix defined by $D_{p,p} = \sum_{p'} W_{p,p'}$. $\mathbf{f}$ denotes the vector of $f(p)$ for all $p \in \mathcal{P}(S)$. Note that the last term depends only on the value of $f$ on the parts in the set $\mathcal{P}(S)$. Then, for any local loss $\mathcal{L}(x, y, f)$,

we immediately have the following Representer Theorem for the semi-supervised structured case where $S$ includes the labeled and the unlabeled data.

$$f_\alpha^*(p) = \sum_{p' \in \mathcal{P}(S)} \alpha_{p'} k(p', p) \tag{10}$$

Substituting (10) into (9) leads to the following optimization problem

$$\alpha^* = \operatorname*{argmin}_\alpha \sum_{i=1}^\ell \mathcal{L}(x^i, y^i, f_\alpha) + \alpha^T Q \alpha, \tag{11}$$

where $Q = \lambda_1 K + \lambda_2 K L K$, $K$ is the matrix of $k(p, p')$ for all $p, p' \in \mathcal{P}(S)$ and $f_\alpha$, as a vector in the space $\mathcal{H}_k$, is a linear function of the vector $\alpha$. Note that (11) applies to any local loss function and if $\mathcal{L}(x, y, f)$ is convex in $f$, as in the case for logistic or hinge loss, then (11) is convex in $\alpha$.

We now have a loss function over labeled data regularized by the $L_2$ norm (wrt the inner product Q), for which we can re-evoke the Representer Theorem. Let $S^\ell$ be the set of labeled inputs $\{x^1, \ldots, x^\ell\}$, $\mathcal{Z}(S^\ell)$ be the set of all pairs $\langle x, y \rangle$ with $x \in \mathcal{X}(S^\ell)$ and $y \in \mathcal{Y}(x)$ and $\mathcal{P}(S^\ell)$ be the set of al parts having nonzero count for some pair in $\mathcal{Z}(S^\ell)$. Let $\delta_p$ be a vector whose $p$th component is 1 and 0 elsewhere. Using the standard orthogonality argument, let $\alpha^*$ decompose into two: the vector in the span of $\gamma_p = \delta_p K Q^{-1}$ for all $p \in \mathcal{P}(S^\ell)$, and the vector in the orthogonal component (under the inner product $Q$).

$$\alpha = \sum_{p \in \mathcal{P}(S^\ell)} \beta_p \gamma_p + \alpha_\perp$$

$\alpha_\perp$ can only increase the quadratic term in the optimization problem. Notice that the first term in (11) depends only on $f_\alpha(p)$ for $p \in \mathcal{P}(S^\ell)$,

$$f_\alpha(p) = \delta_p K \alpha = (\delta_p K Q^{-1}) Q \alpha = \gamma_p Q \alpha.$$

Since $\gamma_p Q \alpha_\perp = 0$, we conclude that the optimal solution to (11) is given by

$$\alpha^* = \sum_{p \in \mathcal{P}(S^\ell)} \beta_p \gamma_p = \beta K Q^{-1}, \tag{12}$$

where $\beta$ is required to be sparse, such that only parts from the labeled data are nonzero. Plugging this into original equations we get

$$\tilde{k}(p, p') = k_p Q^{-1} k_{p'} \tag{13}$$

$$f_\beta(p') = \sum_{p \in \mathcal{P}(S^\ell)} \beta_p \tilde{k}(p, p') \tag{14}$$

$$\beta^* = \operatorname*{argmin}_\beta \mathcal{L}(S^\ell, f_\beta) + \beta^T \tilde{K} \beta \tag{15}$$

where $k_p$ is the vector of $k(p, p')$ for all $p' \in \mathcal{P}(S)$ and $\tilde{K}$ is the matrix of $\tilde{k}(p, p')$ for all $p, p'$ in $\mathcal{P}(S^\ell)$. $\tilde{k}$ is the same as in [13].

We call $\tilde{k}$ the *graph-regularized* kernel, in which unlabeled data points are used to augment the base kernel $k$ wrt the standard graph kernel to take the underlying density on parts into account. This kernel is defined over the complete part space, where as standard graph kernels are restricted to $\mathcal{P}(S)$ only.

Given the graph-regularized kernel, the semi-supervised structured learning problem is reduced to supervised structured learning. Since in semi-supervised learning problems, in general, labeled data points are far fewer than unlabeled data, the dimensionality of the optimization problems is greatly reduced by this reduction.

## 4 Structured Max-Margin Learning

We now investigate optimizing the hinge loss as defined by (6) using graph-regularized kernel $\tilde{k}$. Defining $\gamma^{x,y}$ to be the vector where $\gamma_p^{x,y} = c(p, \langle x, y \rangle)$ is the count of $p$ in $\langle x, y \rangle$, the linear discriminant can be written in matrix notation for $x \in S^\ell$ as

$$F_{f_\beta}(x, y) = \beta^T \tilde{K} \gamma^{x,y}.$$

Then, the optimization problem for margin maximization is

$$\beta^* = \underset{\beta}{\operatorname{argmin}} \min_{\xi} \sum_{i=1}^{l} \xi_i + \beta^T \tilde{K} \beta$$

$$\xi_i \geq \max_{\hat{y} \in \mathcal{Y}(x^i)} \triangle(\hat{y}, y^i) - \beta^T \tilde{K} \left( \gamma^{x^i, y^i} - \gamma^{x^i, \hat{y}} \right) \quad \forall i \leq l.$$

This gives a convex quadratic program over the vectors indexed by $\mathcal{P}(S)$, a polynomial size problem in terms of the size of the structures. Following [2], we replace the convex constraints by linear constraints for all $y \in \mathcal{Y}(x)$ and using Lagrangian duality techniques, we get the following dual Quadratic program:

$$\theta^* = \underset{\theta}{\operatorname{argmin}} \, \theta^T d\!R \, \theta - \Delta^T \theta \tag{16}$$

$$\theta_{(x^i, y)} \geq 0, \sum_{y \in \mathcal{Y}(x)} \theta_{(x^i, y)} = 1, \quad \forall y \in \mathcal{Y}(x^i), \quad \forall i \leq l,$$

where $\Delta$ is a vector of $\triangle(y, \hat{y})$ for all $y \in \mathcal{Y}(x)$ of all labeled observations $x$, $d\!\gamma$ is a matrix whose $(x^i, y)$th column $d\!\gamma_{\cdot, (x^i, y)} = \gamma^{x^i, y^i} - \gamma^{x^i, y}$ and $d\!R = d\!\gamma^T \tilde{K} d\!\gamma$. Due to the sparse structure of the constraint matrix, even though this is an exponential sized QP, the algorithm proposed in [2] is proven to solve (16) to $\eta$ proximity in polynomial time in $\mathcal{P}(S^l)$ and $\frac{1}{\eta}$ [15].

## 5 Semi-Supervised vs Transductive Learning

Since one major contribution of this paper is learning a classifier for structured objects that is defined over the complete part space $\mathcal{P}$, we now examine the differences of semi-supervised and transductive learning in more detail. The most common approach to realize the smoothness assumption is to construct a data dependent kernel $k_S$ derived from the graph Laplacian on a nearest neighbor graph on the labeled and unlabeled input patterns in the sample $S$. Thus, $k_S$ is not defined on observations that are out of the sample. Given $k_S$, one can construct a function $\tilde{f}^*$ on $S$ as

$$\tilde{f}^* = \underset{f \in \mathcal{H}_{k_S}}{\operatorname{argmin}} \sum_{i=1}^{\ell} \mathcal{L}(x^i, y^i, f) + \lambda \|f\|_{k_S}^2. \tag{17}$$

It is well known that kernels can be combined linearly to yield new kernels. This observation in the transductive setting leads to the following optimization problem, when the kernel of the optimization problem is taken to be a linear combination of a graph kernel $k_S$ and a standard kernel $k$ restricted to $\mathcal{P}(S)$.

$$\bar{f}^* = \underset{f \in \mathcal{H}_{(\mu_1 k + \mu_2 k_S)}}{\operatorname{argmin}} \sum_{i=1}^{\ell} \mathcal{L}(x^i, y^i, f) + \lambda \|f\|_{(\mu_1 k + \mu_2 k_S)}^2 \tag{18}$$

A structured semi-supervised algorithm based on (18) has been evaluated in [11]. The kernel is (18) is the weighted mean of $k$ and $k_S$, whereas the graph-regularized

kernel, resulting from weighted mean of two regularizers, is the harmonic mean of $k$ and $k_S$ [16]. An important distinction between $\bar{f}^*$ and $f^*$ in (8), the optimization performed in this paper, is that $\bar{f}^*$ is only defined on $\mathcal{P}(S)$ (only on observations in the training data) while $f^*$ is defined on all of $\mathcal{P}$ and can be used for novel (out of sample) inputs $x$. We note that in general $\mathcal{P}$ is infinite. Out-of-sample extension is already a serious limitation for transductive learning, but it is even more severe in the structured case where parts of $\mathcal{P}$ can be composed of multiple observation tokens.

## 6   Experiments

**Similarity Graph:**   We build the similarity matrix $W$ over $\mathcal{P}(S)$ using K-nearest neighborhood relationship. $W_{p,p'}$ is 0 if $p$ and $p'$ are not in the K-nearest neighborhood of each other or if $p$ and $p'$ are of different types. Otherwise, the similarity is given by a heat kernel. In our applications, the structure is a simple chain, therefore the cliques involved single observation label pairs,

$$W_{p,p'} = \delta(y(u_p), y(u'_{p'}))e^{\frac{\|u_p - u'_{p'}\|^2}{t}}, \tag{19}$$

where $u_p$ denotes the observation part of $p$ and $y(u)$ denotes the labeling of $u$ [1]. In cases where $k(p,p') = W_{p,p'} = 0$ for $p, p'$ of different types, as in our experiments, the Gram matrix $K$ and the Laplacian $L$ can be presented as block diagonal matrices, which significantly reduces the computational complexity, the computation of $Q^{-1}$ in particular.

**Applications:**   We performed experiments using a simple chain model for pitch accent (PA) prediction and OCR. In PA prediction, $\mathcal{Y}(x) = \{0,1\}^T$ with $T = |x|$ and $x_t \in \Re^{31}, \forall t$. In OCR, $x_t \in \{0,1\}^{128}$ and $|\Sigma| = 15$.

We ran experiments comparing semi-supervised structured (referred as STR) and unstructured (referred as SVM) max-margin optimization. For both SVM and STR, we used RBF kernel as the base kernel $k_o$ in (4) and a 5-nearest neighbor graph to construct the Laplacian.

| PA | U:0 | U:80 | U:0 | U:80 | U:200 |
|---|---|---|---|---|---|
| SVM | 65.92 | 68.83 | 70.34 | 71.27 | 73.68 |
|  | - | 69.94 | - | 72.00 | 73.11 |
| STR | 65.81 | 70.28 | 72.15 | 74.92 | 76.37 |
|  | - | 70.72 | - | 75.66 | 77.45 |

Table 1: Per-label accuracy for Pitch Accent.

We chose the width of the RBF kernel by cross-validation on SVM and used the same value for STR. Following [3], we fixed $\lambda_1 : \lambda_2$ ratio at $1 : 9$. We report the average results of experiments with 5 random selection of labeled sequences in Table 1 and 2, with number of labeled sequences 4 on the left side of Table 1, 40 on the right side, and 10 in Table 2. We varied the number of unlabeled sequences and reported the per-label accuracy of test sequences (on top of each cell) and of unlabeled sequences (bottom) (when $U > 0$). The results in pitch accent prediction shows the advantage of a sequence model over a non-structured

model, where STR consistently performs better than SVM. We also observe the usefulness of unlabeled data both in the structured and unstructured models, where as $U$ increases, so does the accuracy. The improvement from unlabeled data and from structured classification can be considered as additive. The small difference between the accuracy of in-sample unlabeled data and the test data indicates the natural extension of our framework to new data points.

In OCR, on the other hand, STR does not improve over SVM. Even though unlabeled data improves accuracy, performing sequence classification is not helpful due to the sparsity of structural information. Since $|\Sigma| = 15$ and there are only 10 labeled sequences with average length 8.3, the statistics of label-label dependency is quite noisy.

| OCR | U:0 | U:412 |
|-----|------|-------|
| SVM | 43.62 | 49.96 |
|     | -    | 47.56 |
| STR | 49.25 | 49.91 |
|     | -    | 49.65 |

Table 2: OCR

## 7    Conclusions

We presented a discriminative approach to semi-supervised learning of structured and inter-dependent response variables. In this framework, we derived a maximum margin formulation and presented experiments for a simple chain model. Our approach naturally extends to the classification of unobserved structured inputs and this is supported by our empirical results which showed similar accuracy on in-sample unlabeled data and out-of-sample test data.

## Footnotes

[1]For more complicated parts, different measures can apply. For example, in sequence classification, if the classifier is evaluated wrt the correctly classified individual labels in the sequence, W can be s. t. $W_{p,p'} = \sum_{u \in p, u' \in p'} \delta(y(u), y(u'))\tilde{s}(u, u')$ where $\tilde{s}$ denotes some similarity measure such as the heat kernel. If the evaluation is over segments of the sequence, the similarity can be $W_{p,p'} = \delta(y(p), y'(p')) \sum_{u \in p, u' \in p'} \tilde{s}(u, u')$ where $y(p)$ denotes all the label nodes in the part $p$.

## References

[1] Y. Altun, T. Hofmann, and A. Smola. Gaussian process classification for segmenting and annotating sequences. In *ICML*, 2004.

[2] Y. Altun, I. Tsochantaridis, and T. Hofmann. Hidden markov support vector machines. In *ICML*, 2003.

[3] M. Belkin, P. Niyogi, and V. Sindhwani. Manifold regularization: a geometric framework for learning from examples. Technical Report 06, UChicago CS, 2004.

[4] Avrim Blum and Tom Mitchell. Combining labeled and unlabeled data with co-training. In *COLT*, 1998.

[5] U. Brefeld, C. Büscher, and T. Scheffer. Multi-view discriminative sequential learning. In *(ECML)*, 2005.

[6] O. Chappelle, J. Weston, and B. Scholkopf. Cluster kernels for semi-supervised learning. In *(NIPS)*, 2002.

[7] M. Collins and N.l Duffy. Convolution kernels for natural language. In *(NIPS)*, 2001.

[8] Olivier Delalleau, Yoshua Bengio, and Nicolas Le Roux. Efficient non-parametric function induction in semi-supervised learning. In *Proceedings of AISTAT*, 2005.

[9] Thorsten Joachims. Transductive inference for text classification using support vector machines. In *(ICML)*, pages 200–209, 1999.

[10] G. Kimeldorf and G. Wahba. Some results on tchebychean spline functions. *Journal of Mathematics Analysis and Applications*, 33:82–95, 1971.

[11] John Lafferty, Yan Liu, and Xiaojin Zhu. Kernel conditional random fields: Representation, clique selection, and semi-supervised learning. In *(ICML)*, 2004.

[12] K. Nigam, A. K. McCallum, S. Thrun, and T. M. Mitchell. Learning to classify text from labeled and unlabeled documents. In *Proceedings of AAAI-98*, pages 792–799, Madison, US, 1998.

[13] V. Sindhwani, P. Niyogi, and M. Belkin. Beyond the point cloud: from transductive to semi-supervised learning. In *(ICML)*, 2005.

[14] B. Taskar, C. Guestrin, and D. Koller. Max-margin markov networks. In *NIPS*, 2004.

[15] I. Tsochantaridis, T. Hofmann, T. Joachims, and Y. Altun. Support vector machine learning for interdependent and structured output spaces. In *(ICML)*, 2004.

[16] T. Zhang. personal communication.
